# Bayesian estimation of orientation preference maps

**Jakob H. Macke**
MPI for Biological Cybernetics
and University of Tübingen
Computational Vision and Neuroscience
Spemannstrasse 41, 72076 Tübingen
jakob@tuebingen.mpg.de

**Sebastian Gerwinn**
MPI for Biological Cybernetics
and University of Tübingen
Computational Vision and Neuroscience
Spemannstrasse 41, 72076 Tübingen
sgerwinn@tuebingen.mpg.de

**Leonard E. White**
Duke Institute for Brain Sciences
Duke University
Durham, NC 27705, USA
white033@mc.duke.edu

**Matthias Kaschube**
Lewis-Sigler Institute for Integrative Genomics
and Department of Physics
Princeton University
Princeton, NJ 08544, USA
kaschube@princeton.edu

**Matthias Bethge**
MPI for Biological Cybernetics
and University of Tübingen
Computational Vision and Neuroscience Group
Spemannstrasse 41,
72076 Tübingen
mbethge@tuebingen.mpg.de

## Abstract

Imaging techniques such as optical imaging of intrinsic signals, 2-photon calcium imaging and voltage sensitive dye imaging can be used to measure the functional organization of visual cortex across different spatial and temporal scales. Here, we present Bayesian methods based on Gaussian processes for extracting topographic maps from functional imaging data. In particular, we focus on the estimation of orientation preference maps (OPMs) from intrinsic signal imaging data. We model the underlying map as a bivariate Gaussian process, with a prior covariance function that reflects known properties of OPMs, and a noise covariance adjusted to the data. The posterior mean can be interpreted as an optimally smoothed estimate of the map, and can be used for model based interpolations of the map from sparse measurements. By sampling from the posterior distribution, we can get error bars on statistical properties such as preferred orientations, pinwheel locations or pinwheel counts. Finally, the use of an explicit probabilistic model facilitates interpretation of parameters and quantitative model comparisons. We demonstrate our model both on simulated data and on intrinsic signaling data from ferret visual cortex.

## 1 Introduction

Neurons in the visual cortex of primates and many other mammals are organized according to their tuning properties. The most prominent example of such a topographic organization is the layout of neurons according to their preferred orientation, the orientation preference map (OPM) [1, 2, e.g.]. The statistical structure of OPMs [3, 4] and other topographic maps has been the focus of extensive

research, as have been the relationships between different maps [5]. Orientation preference maps can be measured using optical imaging of intrinsic signals, voltage sensitive dye imaging, functional magnetic resonance imaging [6], or 2-photon calcium imaging [2, 7]. For most of these methods the signal-to-noise ratio is low, i.e. the stimulus specific part of the response is small compared to non-specific background fluctuations. Therefore, statistical pre-processing of the data is required in order to extract topographic maps from the raw experimental data. Here, we propose to use Gaussian process methods [8] for estimating topographic maps from noisy imaging data. While we will focus on the case of OPMs, the methods used will be applicable more generally.

The most common analysis method for intrisic signaling data is to average the data within each stimulus condition, and report differences between conditions. In the case of OPMs, this amounts to estimating the preferred orientation at each pixel by vector averaging the different stimulus orientations weighted according to the evoked responses. In a second step, spatial bandpass filtering is usually applied in order to obtain smoother maps. One disadvantage of this approach is that the frequency characteristics of the bandpass filters are free parameters which are often set ad-hoc, and may have a substantial impact on the statistics of the obtained map [9, 10]. In addition, the approach ignores the effect of anisotropic and correlated noise [11, 10], which might result in artifacts.

Methods aimed at overcoming these limitations include analysis techniques based on principal component analysis, linear discriminant analysis, oriented PCA [12] (and extensions thereof [11]) as well as variants of independent component analysis [9]. Finally, paradigms employing periodically changing stimuli [13, 14] use differences in their temporal characteristics to separate signal and noise components. These methods have in common that they do not make any parametric assumptions about the relationship between stimulus and response, between different stimuli, or about the smoothness of the maps. Rather, they attempt to find 'good' maps by searching for filters which are maximally discriminative between different stimulus conditions. In particular, they differ from the classical approach in that they do not assume the noise to be isotropic and uncorrelated, but make it hard to incorporate prior knowledge about the structure of maps, and can therefore be data-intensive. Here, we attempt to combine the strengths of the classical and discriminative models by combining prior knowledge about maps with flexible noise models into a common probabilistic model.

We encode prior knowledge about the statistical structure of OPMs in the covariance function of a Gaussian Process prior over maps. By combining the prior with the data through an explicit generative model of the measurement process, we obtain a posterior distribution over maps. Compared to previously proposed methods for analyzing multivariate imaging methods, the GP approach has a number of advantages:

- Optimal smoothing: The mean of the posterior distribution can be interpreted as an optimally smoothed map. The filtering is adaptive, i.e. it will adjust to the amount and quality of the data observed at any particular location.

- Non-isotropic and correlated noise: In contrast to the standard smoothing approach, noise with correlations across pixels as well as non-constant variances can be modelled.

- Interpolations: The model returns an estimate of the preferred orientation at any location, not only at those at which measurements were obtained. This can be used, e.g., for artifact removal, or for inferring maps from multi-electrode recordings.

- Explicit probabilistic model: The use of an explicit, generative model of the data facilitates both the interpretation and setting of parameters quantitative model comparisons.

- Model based uncertainty estimates: The posterior variances at each pixel can be used to compute point-wise error bars at each pixel location [9, 11]. By sampling from the posterior (using the full posterior covariance), we can also get error bars on topological or global properties of the map, such as pinwheel counts or locations.

Mathematically speaking, we are interested in inferring a vector field (the 2-dimensional vector encoding preferred orientation) across the cortical surface from noisy measurements. Related problems have been studied in spatial statistics, e.g. in the estimation of wind-fields in geo-statistics [15], where GP methods for this problem are often referred to as *co-kriging* methods [16, 17].

## 2 Methods

### 2.1 Encoding Model

We model an imaging experiment, where at each of $N$ trials, the activity at $n$ pixels is measured. The response $r_i(x)$ at trial $i$ to a stimulus parameterised by $V_i$ is given by

$$r_i(x) = \sum_{k=1}^{d} v_{ki} m_k(x) + \epsilon_i(x) = v_i^\top m_k(x) + \epsilon_i(x), \tag{1}$$

i.e. the mean response at each pixel is modelled to be a linear function of some stimulus parameters $v_{ki}$.

This can be written compactly as $\mathbf{r_i} = M v_i + \varepsilon_i$ or $\mathbf{r_i} = V_i^\top \mathbf{m} + \varepsilon_i$. Here, $\mathbf{r_i}$ and $\varepsilon_i$ are $n$-dimensional vectors, $M$ is an $n \times d$ dimensional matrix, $V_i = v_i \otimes \mathbb{I}_n$, $\otimes$ is the Kronecker-product and $\mathbf{m} = \text{vec}(M)$ is an $nd$-dimensional vector.

We refer to the coefficients $m_k(x)$ as *feature maps*, as they indicate the selectivity of pixel $x$ to stimulus feature $k$. In the specific case of modelling an orientation preference map, we have $d = 2$ and $v_i = (\cos(2\theta_i), \sin(2\theta_i))^\top$. Then, the argument of the complex number $m'(x) = m_1(x) + \mathbf{i}m_2(x)$ is the *preferred orientation* at location $x$, whereas the absolute value of $m'(x)$ is a measure of its selectivity. While this approach assumes cosine-tuning curves at each measurement location, it can be generalized to arbitrary tuning curves by including terms corresponding to cosines with different frequencies.

We assume that the noise-residuals $\varepsilon$ are normally distributed with covariance $\Sigma_\epsilon$, and a Gaussian prior with covariance $K_m$ for the feature map vector $\mathbf{m}$. Then, the posterior distribution over $\mathbf{m}$ is Gaussian with posterior covariance $\Sigma_{\text{post}}$ and mean $\mu_{\text{post}}$:

$$\Sigma_{\text{post}}^{-1} = K_m^{-1} + \left( \sum_i v_i v_i^\top \right) \otimes \Sigma_\epsilon^{-1} \tag{2}$$

$$\mu_{\text{post}} = \Sigma_{\text{post}} \left( \sum_i V_j \Sigma_\epsilon^{-1} \mathbf{r}_i \right)$$

$$= \Sigma_{\text{post}} \left( \mathbb{I}_d \otimes \Sigma_\epsilon^{-1} \right) \sum_i v_i \otimes \mathbf{r}_i \tag{3}$$

We note that the posterior covariance will have block structure provided that the prior covariance $K_m$ has block structure, i.e. if different feature maps are statistically independent *a priori*, and the stimuli are un-correlated on average, i.e. $\sum_i v_i v_i^\top = D_v$ is diagonal. Hence, inference for different maps 'de-couples', and we do not have to store the full joint covariance over all $d$ maps.

### 2.2 Choosing a prior

We need to specify the covariance function $K(m(x), m(x'))$ of the prior distribution over maps. As cortical maps, and in particular orientation preference maps, have been studied extensively in the past [5], we actually have prior *knowledge* (rather than just prior assumptions) to guide the choice of a prior. It is known that orientation preference maps are smooth [2] and that they have a semi-periodic structure of regularly spaced columns. Hence, filtering white noise with appropriately chosen filters [18] yields maps which visually look like measured OPMs (see Fig. 1). While it is known that real OPMs differ from Gaussian random fields in their higher order statistics [3], use of a Gaussian prior can be motivated by the maximum entropy principle: We assume a prior with minimal higher-order correlations, with the goal of inferring them from the experimental data [3]. For simplicity, we take the prior to be isotropic, i.e. not to favour any direction over others. (For real maps, there is a slight anisotropy [19]).

We assume that each prior sample is generated by convolving a two-dimensional Gaussian white-noise image with a Difference-of-Gaussians filter $f(x) = \sum_{k=1}^{2} \frac{\alpha_k}{2\pi\sigma_k^2} \exp\left( -\frac{1}{2} \frac{x^2}{\sigma_k^2} \right)$, $\alpha_1 = -\alpha_2$, and $\sigma_2 = 2\sigma_1$. This will result in a prior which is uncorrelated in the different maps component, i.e.

$\text{Cov}(m_1(x), m_2(x')) = 0$, and a stationary covariance function given by

$$K_c(\tau) = K_c(\|x - x'\|) = \text{Cov}(m_1(x), m_1(x'))$$

$$= \sum_{k,l=1}^{2} \frac{\alpha_k \alpha_l}{2\pi(\sigma_k^2 + \sigma_l^2)} \exp\left(-\frac{1}{2}\left(\frac{\tau^2}{\sigma_k^2 + \sigma_l^2}\right)\right). \qquad (4)$$

Then, the prior covariance matrix $K_m$ can be written as $K_m = \mathbb{I}_c \otimes K_c$. This prior has two hyper-parameters, namely the absolute magnitude $\alpha_1$ and the kernel width $\sigma_1$. In principle, optimization of the marginal likelihood can be used to set hyper-parameters. In practice, it turned out to be computationally more efficient to select them by matching the radial component of the empirically observed auto-correlation function of the map [16], see Fig. 1 B).

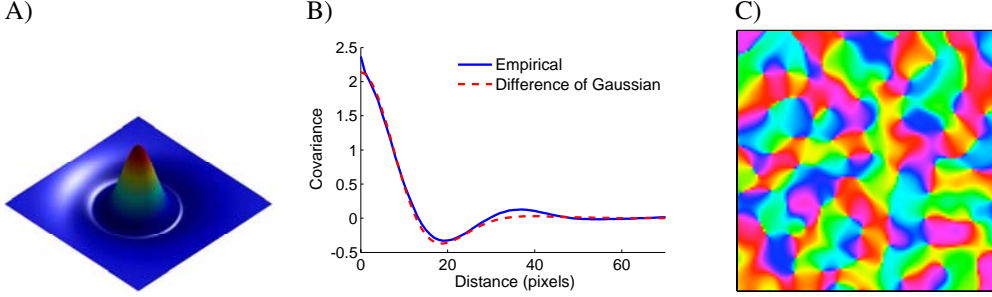

Figure 1: **Prior covariance: A)** Covariance function derived from the Difference-of-Gaussians. **B)** Radial component of prior covariance function and of covariance of raw data. **C** Angle-map of one sample from the prior, with $\sigma_1 = 4$. Each color corresponds to an angle in $[0, 180°]$.

## 2.3 Approximate inference

The formulas for the posterior mean and covariance involve covariance matrices over all pixels. On a map of size $n_x \times n_y$, there are $n = n_x \times n_y$ pixels, so we would have to store and compute with matrices of size $n \times n$, which would limit this approach to maps of relatively small size. A number of approximation techniques have been proposed to make large scale inference feasible in models with Gaussian process priors (see [8] for an overview). Here, we utilize the fact that the spectrum of eigenvalues drops off quickly for many kernel functions [20, 21], including the Difference-of-Gaussians used here. This means that the covariance matrix $K_c$ can be approximated well by a low rank matrix product $K_c \approx GG^\top$, where $G$ is of size $n \times q$, $q \ll n$ (see [17] for a related idea). To find $G$, we perform an incomplete Cholesky factorization on the matrix $K_c$. This can be done without having to store $K_c$ in memory explicitly.

In this case, the posterior covariance can be calculated without ever having to store (or even invert) the full prior covariance:

$$\Sigma_{\text{post}} = \mathbb{I}_d \otimes \left(K_c - \beta^{-1}K_c\left(\Sigma_\epsilon^{-1} - \Sigma_\epsilon^{-1}G\left(\beta\mathbb{I}_q + G^\top\Sigma_\epsilon^{-1}G\right)^{-1}G^\top\Sigma_\epsilon^{-1}\right)K_c\right), \qquad (5)$$

where $\beta = 2/N$. We restrict the form of the noise covariance either to be diagonal (i.e. assume uncorrelated noise), or more generally to be of the form $\Sigma_\epsilon = D_\epsilon + G_\epsilon R_\epsilon G_\epsilon^\top$. Here, $G_\epsilon$ is of size $n \times q_\epsilon$, $q_\epsilon \ll n$, and $D_\epsilon$ is a diagonal matrix. In other words, the functional form of the covariance matrix is assumed to be the same as in *factor analysis* models [22, 23]: The low rank term $G_\epsilon$ models correlation across pixels, whereas the diagonal matrix $D_\epsilon$ models independent noise. We assume this model to regularize the noise covariance to ensure that the noise covariance has full rank even when the number of data-points is less than the number of pixels [22]. The matrices $G_\epsilon$ and $D_\epsilon$ can be fit using expectation maximization without ever having to calculate the full noise covariance across all pixels. We initialize the noise covariance by calculating the noise variances for each stimulus condition, and averaging this initial estimate across stimulus conditions. We iterate between calculating the posterior mean (using the current estimate of $\Sigma_\epsilon$), and obtaining a point-estimate of the most likely noise covariance given the mean [24]. In all cases, a very small number of iterations lead to convergence.

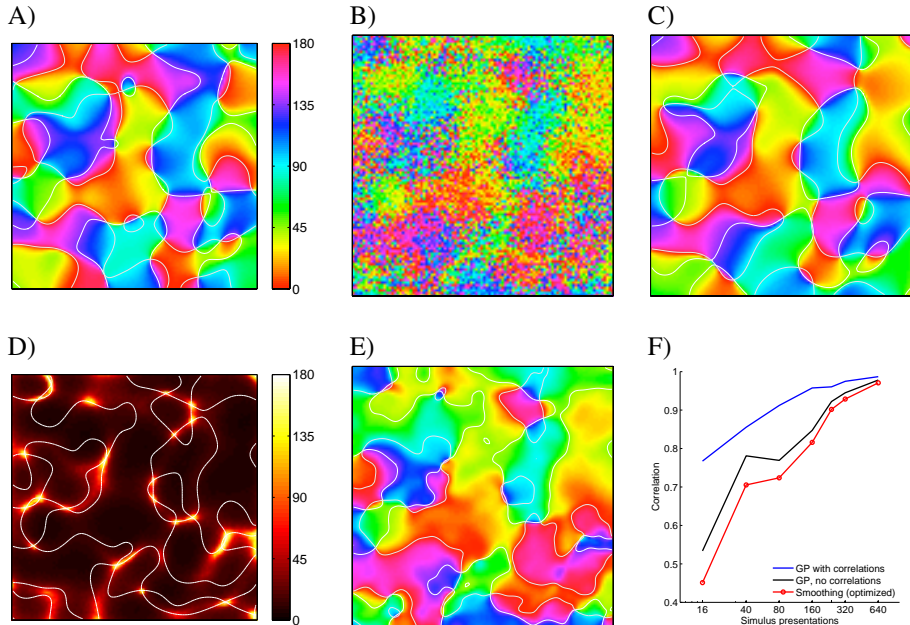

Figure 2: **Illustration on synthetic data: A)** Ground truth map used to generate the data. **B)** Raw map, estimated using 10 trials of each direction. **C)** GP-reconstruction of the map. **D)** Posterior variance of GP, visualized as size of $95\%$ confidence intervals on preferred orientations. Superimposed are the zero-crossings of the GP map. **E)** Reconstruction by smoothing with fixed Gaussian filter, filter-width optimized by maximizing correlation with ground truth. **F)** Reconstruction performance as a function of stimulus presentations used, for GP with noise-correlations, GP without noise-correlations, and simple smoothing.

## 3 Results

### 3.1 Illustration on synthetic data

To illustrate the ability of our method to recover maps from noisy recordings, we generated a synthetic map (a sample from the prior distribution, 'true map', see Fig. 2 A), and simulated responses to each of 8 different oriented gratings by sampling from the likelihood (1). The parameters were chosen to be roughly comparable with the experimental data (see below). We reconstructed the map using our GP method (low rank approximation of rank $q = 1600$, noise correlations of rank $q_\epsilon = 5$) on data sets of different sizes ($N = 8 * (2, 5, 10, 20, 30, 40, 80)$). Figure 2 C) shows the angular components of the posterior mean of the GP, our reconstruction of the map. We use the posterior variances to also calculate a pointwise $95\%$ confidence interval on the preferred orientation at each location, shown in Fig. 2 D). As expected, the confidence intervals are biggest near pinwheels, where the orientation selectivity of pixels is low, and therefore the preferred orientation is not well defined.

To evaluate the performance of the model, we quantified its reconstruction performance by computing the correlation coefficient of the posterior mean and the true map, each represented as a long vector with $2n$ elements. We compared the GP map against a map obtained by filtering the raw map (Fig. 2 B) with a Gaussian kernel (Fig. 2 D), where the kernel width was chosen by maximizing the similarity with the 'true map'. This yields an optimistic estimate of the performance of the smoothed map, as setting the optimal filter-size requires access to the ground truth. We can see that the GP map converges to the true map more quickly than the smoothed map (Fig. 2 F). For example, using 16 stimulus presentations, the smoothed map has a correlation with the ground truth of $0.45$, whereas the correlation of the GP map is $0.77$. For the simple smoothing method, about 120 presentations would be required to achieve this performance level. When we ignore noise-correlations (i.e. assume $\Sigma_\epsilon$ to be diagonal), GP still outperforms simple smoothing, although by a much smaller amount (Fig. 2 F).

## 3.2 Application to data from ferret visual cortex

To see how well the method works on real data, we used it to analyze data from an intrinsic signal optical imaging experiment. The central portion of the visuotopic map in visual areas V1 and V2 of an anesthetized ferret was imaged with red light while square wave gratings (spatial frequency 0.1 cycles/degree) were presented on a screen. Gratings were presented in 4 different orientations ($0°$, $45°$, $90°$ and $135°$), and moving along one of the two directions orthogonal to its orientation (temporal frequency 3.2Hz). Each of the 8 possible directions was presented 100 times in a pseudo-random order for a duration of 5 seconds each, with an interstimulus interval of 8 seconds. Intrinsic signals were collected using a digital camera with pixel-size $30\mu m$. The response $\mathbf{r}_i$ was taken to be the average activity in a 5 second window relative to baseline Each response vector $\mathbf{r}_i$ was normalized to have mean 0 and standard deviation 1, no spatial filtering was performed. For all analyses in this paper, we concentrated on a region of size 100 by 100 pixels. The large data set with a total of 800 stimulus presentations made it possible to quantify the performance of our model by comparing it to unsmoothed maps. Figure 3 A) shows the map estimated by vector averaging all 800 presentations, without any smoothing. However, the GP method itself is designed to also work robustly on smaller data sets, and we are primarily interested in its performance in estimating maps using only few stimulus presentations.

## 3.3 Bayesian estimation of orientation preference maps

For real measured data, we do not know ground truth to estimate the performance of our model. Therefore, we used $5\%$ of the data for estimating the map, and compared this map with the (un-smoothed) map estimated on the other $95\%$ of data, which served as our proxy for ground truth. As above, we compared the GP map against one obtained by smoothing with a Gaussian kernel, where the kernel width of the smoothing kernel was chosen by maximizing its correlation with (our proxy for) the ground truth. The GP map outperformed the smoothing map consistently: For 18 out of 20 different splits into training and test data, the correlation of the GP map was higher ($p = 2 \times 10^{-4}$, average correlations $c = 0.84 \pm 0.01$ for GP, $c = 0.79 \pm 0.015$ for smoothing). The same held true when we smoothed maps with a Difference of Gaussians filter rather than a Gaussian (19 out of 20, average correlation $c = 0.71 \pm 0.08$).

A)            B)            C)

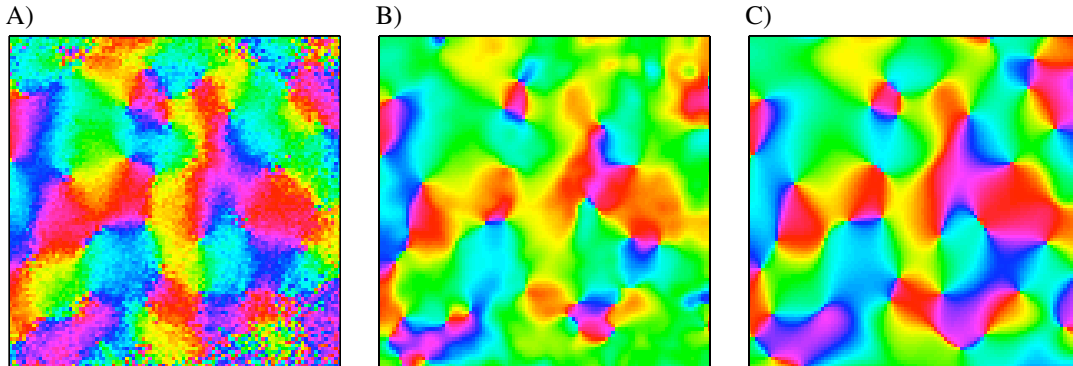

Figure 3: **OPMs in ferret V1 A)** Raw map, estimated from 720 out of 800 stimuli. **B)** Smoothed map estimated from other 80 stimuli, filter width obtained by maximizing the correlation to map A. **C)** GP reconstruction of map. The GP has a correlation with the map shown in A) of $0.87$, the performance of the smoothed map is $0.74$.

One of the strengths of the GP model is that the filter-parameters are inferred by the model, and do not have to be set ad-hoc. The analysis above shows that, even if when optimized the filter-width for smoothing (which would not be possible in a real experiment), the GP still outperforms the approach of smoothing with a Gaussian window. In addition, it is important to keep in mind that using the posterior mean as a clean estimate of the map is only one feature of our model. In the following, we will use the GP model to optimally interpolate a sparsely sampled map, and to the posterior distribution to obtain error bars over the pinwheel-counts and locations of the map.

## 3.4    Interpolating the map

The posterior mean $\mu(x)$ of the model can be evaluated for any $x$. This makes it possible to extend the map to locations at which no data was recorded. We envisage this to be useful in two kinds of applications: First, if the measurement is corrupted in some pixels (e.g. because of a vessel artifact), we attempt to recover the map in this region by model-based interpolation. We explored this scenario by cutting out a region of the map described above (inside of ellipse in Fig. 4 A), and using the GP to fill in the map. The correlation between the true map and the GP map in the filled-in region was 0.77. As before, we compared to smoothing with a Gaussian filter, for which the correlation was 0.59.

In addition, multi-electrode arrays [25] can be used to measure neural activity at multiple locations simultaneously. Provided that the electrode spacing is small enough, it should be possible to reconstruct at least a rough estimate of the map from such discrete measurements. We simulated a multi-electrode recording by only using the measured activity at 49 pixel locations which were chosen to be spaced $400\mu m$ apart. Then, we attempted to infer the full map using only these 49 measurements, and our prior knowledge about OPMs encoded in the prior covariance. The reconstruction is shown in Fig. 4 C. As before, the GP map outperforms the smoothing approach ($c = 0.78$ vs. $c = 0.81$). Discriminative analysis methods for imaging data can not be used for such interpolations.

A)        B)        C)        D)

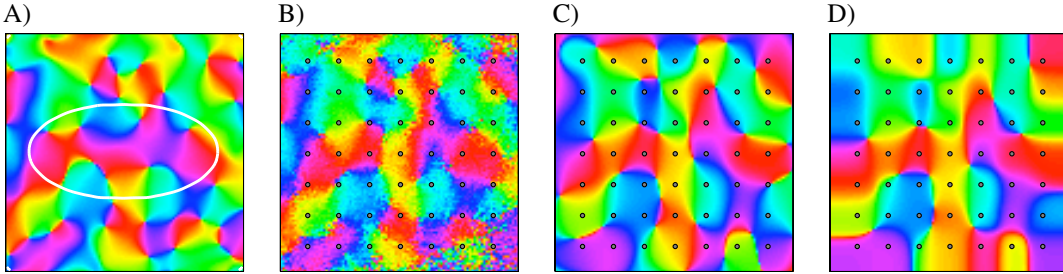

Figure 4: **Interpolations: A)** Filling in: The region inside the white ellipse was reconstructed by the GP using only the data outside the ellipse. **B)** Map estimated from all 800 stimulus presentations, with 'electrode locations' superimposed. **C)** GP-reconstruction of the map, estimated only from the 49 pixels colored in in gray in B). **D)** Smoothing reconstruction of the map.

## 3.5    Posterior uncertainty

As both our prior and the likelihood are Gaussian, the posterior distribution is also Gaussian, with mean $\mu_{\text{post}}$ and covariance $\Sigma_{\text{post}}$. By sampling from this posterior distribution, we can get error bars not only on the preferred orientations in *individual* pixels (as we did for Fig. 2 D), but also for *global* properties of the map. For example, the location [10] and total number [3, 4] of pinwheels (singularities at which both map components vanish) has received considerable attention in the past. Figure 5 A) and B) shows two samples from the posterior distribution, which differ both in their pinwheel locations and counts (A: 39, B: 28, C:31). To evaluate our certainty in the pinwheel locations, we calculate a two-dimensional histogram of pinwheel locations across samples (Fig. 5 D and E). One can see that the histogram gets more peaked with increasing data-set size. We illustrate this effect by calculating the entropy of the (slightly smoothed) histograms, which seems to keep decreasing for larger data-set sizes, indicating that we are more confident in the exact locations of the pinwheels.

## 4    Discussion

We introduced Gaussian process methods for estimating orientation preference maps from noisy imaging data. By integrating prior knowledge about the spatial structure of OPMs with a flexible noise model, we aimed to combine the strengths of classical analysis methods with discriminative approaches. While we focused on the analysis of intrinsic signal imaging data, our methods are expected to be also applicable to other kinds of imaging data. For example, functional magnetic

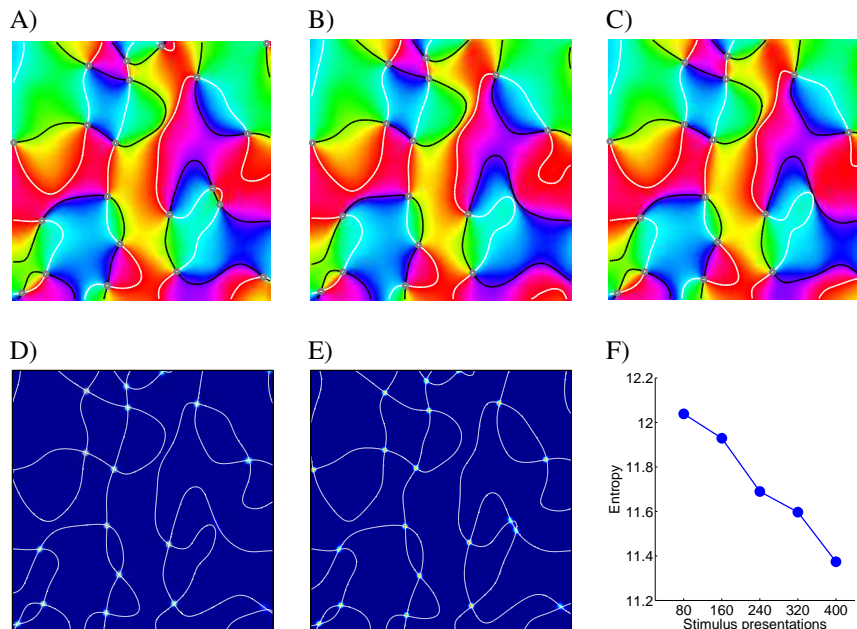

Figure 5: **Posterior uncertainty: A B C)** Three samples from the posterior distribution, using 80 stimuli (zoomed in for better visibility). **D E)** Density-plot of pinwheel locations when map is estimated with 40 and 800 stimuli, respectively. **F)** Entropy of pinwheel-density as a measure of confidence in the pinwheel locations.

resonance imaging is widely used as a non-invasive means of measuring brain activity, and has been reported to be able to estimate orientation preference maps in human subjects [6].

In contrast to previously used analysis methods for intrinsic signal imaging, ours is based on a generative model of the data. This can be useful for quantitative model comparisons, and for investigating the coding properties of the map. For example, it can be used to investigate the relative impact of different model-properties on decoding performance. We assumed a GP prior over maps, i.e. assumed the higher-order correlations of the maps to be minimal. However, it is known that the statistical structure of OPMs shows systematic deviations from Gaussian random fields [3, 4], which implies that there could be room for improvement in the definition of the prior. For example, using priors which are sparse [26] (in an appropriately chosen basis) could lead to superior reconstruction ability, and facilitate reconstructions which go beyond the auto-correlation length of the GP-prior [27]. Finally, one could use generalized linear models rather than a Gaussian noise model [26, 28]. However, it is unclear how general noise correlation structures can be integrated in these models in a flexible manner, and whether the additional complexity of using a more involved noise model would lead to a substantial increase in performance.

## Acknowledgements

This work is supported by the German Ministry of Education, Science, Research and Technology through the Bernstein award to MB (BMBF; FKZ: 01GQ0601), the Werner-Reichardt Centre for Integrative Neuroscience Tübingen, and the Max Planck Society.

## References

[1] G G Blasdel and G Salama. Voltage-sensitive dyes reveal a modular organization in monkey striate cortex. *Nature*, 321(6070):579–85, Jan 1986.

[2] Kenichi Ohki, Sooyoung Chung, Yeang H Ch'ng, Prakash Kara, and R Clay Reid. Functional imaging with cellular resolution reveals precise micro-architecture in visual cortex. *Nature*, 433(7026):597–603, 2005.

[3] F Wolf and T Geisel. Spontaneous pinwheel annihilation during visual development. *Nature*, 395(6697):73–8, 1998.

[4] M. Kaschube, M. Schnabel, and F. Wolf. Self-organization and the selection of pinwheel density in visual cortical development. *New Journal of Physics*, 10(1):015009, 2008.

[5] Naoum P Issa, Ari Rosenberg, and T Robert Husson. Models and measurements of functional maps in v1. *J Neurophysiol*, 99(6):2745–2754, 2008.

[6] Essa Yacoub, Noam Harel, and Kâmil Ugurbil. High-field fmri unveils orientation columns in humans. *P Natl Acad Sci Usa*, 105(30):10607–12, Jul 2008.

[7] Ye Li, Stephen D Van Hooser, Mark Mazurek, Leonard E White, and David Fitzpatrick. Experience with moving visual stimuli drives the early development of cortical direction selectivity. *Nature*, 456(7224):952–6, Dec 2008.

[8] C.E. Rasmussen and C.K.I. Williams. *Gaussian processes for machine learning*. Springer, 2006.

[9] M Stetter, I Schiessl, T Otto, F Sengpiel, M Hübener, T Bonhoeffer, and K Obermayer. Principal component analysis and blind separation of sources for optical imaging of intrinsic signals. *Neuroimage*, 11(5 Pt 1):482–90, May 2000.

[10] Jonathan R Polimeni, Domhnull Granquist-Fraser, Richard J Wood, and Eric L Schwartz. Physical limits to spatial resolution of optical recording: clarifying the spatial structure of cortical hypercolumns. *Proc Natl Acad Sci U S A*, 102(11):4158–4163, 2005 Mar 15.

[11] T. Yokoo, BW Knight, and L. Sirovich. An optimization approach to signal extraction from noisy multivariate data. *Neuroimage*, 14(6):1309–1326, 2001.

[12] R Everson, B W Knight, and L Sirovich. Separating spatially distributed response to stimulation from background. i. optical imaging. *Biological cybernetics*, 77(6):407–17, Dec 1997.

[13] Valery A Kalatsky and Michael P Stryker. New paradigm for optical imaging: temporally encoded maps of intrinsic signal. *Neuron*, 38(4):529–545, 2003 May 22.

[14] A Sornborger, C Sailstad, E Kaplan, and L Sirovich. Spatiotemporal analysis of optical imaging data. *Neuroimage*, 18(3):610–21, Mar 2003.

[15] D. Cornford, L. Csato, D.J. Evans, and M. Opper. Bayesian analysis of the scatterometer wind retrieval inverse problem: some new approaches. *Journal of the Royal Statistical Society. Series B, Statistical Methodology*, pages 609–652, 2004.

[16] N. Cressie. Statistics for spatial data. *Terra Nova*, 4(5):613–617, 1992.

[17] N. Cressie and G. Johannesson. Fixed rank kriging for very large spatial data sets. *Journal of the Royal Statistical Society: Series B (Statistical Methodology)*, 70(1):209–226, 2008.

[18] A S Rojer and E L Schwartz. Cat and monkey cortical columnar patterns modeled by bandpass-filtered 2d white noise. *Biol Cybern*, 62(5):381–391, 1990.

[19] D M Coppola, L E White, D Fitzpatrick, and D Purves. Unequal representation of cardinal and oblique contours in ferret visual cortex. *P Natl Acad Sci Usa*, 95(5):2621–3, Mar 1998.

[20] Francis R Bach and Michael I Jordan. Kernel independent component analysis. *Journal of Machine Learning Research*, 3:1:48, 2002.

[21] C. Williams and M. Seeger. Using the Nystrom method to speed up kernel machines. In *International Conference on Machine Learning*, volume 17, 2000.

[22] Donald Robertson and James Symons. Maximum likelihood factor analysis with rank-deficient sample covariance matrices. *J. Multivar. Anal.*, 98(4):813–828, 2007.

[23] Byron M Yu, John P Cunningham, Gopal Santhanam, Stephen I Ryu, Krishna V Shenoy, and Maneesh Sahani. Gaussian-process factor analysis for low-dimensional single-trial analysis of neural population activity. *J Neurophysiol*, 102(1):614–635, 2009 Jul.

[24] K. Kersting, C. Plagemann, P. Pfaff, and W. Burgard. Most likely heteroscedastic Gaussian process regression. In *Proceedings of the 24th international conference on Machine learning*, pages 393–400. ACM New York, NY, USA, 2007.

[25] Ian Nauhaus, Andrea Benucci, Matteo Carandini, and Dario L Ringach. Neuronal selectivity and local map structure in visual cortex. *Neuron*, 57(5):673–679, 2008 Mar 13.

[26] H. Nickisch and M. Seeger. Convex variational bayesian inference for large scale generalized linear models. In *International Conference on Machine Learning*, 2009.

[27] F. Wolf, K. Pawelzik, T. Geisel, DS Kim, and T. Bonhoeffer. Optimal smoothness of orientation preference maps. *Network: Computation in Neural SystemsComputation in neurons and neural systems*, pages 97–101, 1994.

[28] K. Rahnama Rad and L. Paninski. Efficient estimation of two-dimensional firing rate surfaces via gaussian process methods. *Network: Computation in Neural Systems*, under review, 2009.

